# Orientation contrast sensitivity from long-range interactions in visual cortex

**Klaus R. Pawelzik, Udo Ernst, Fred Wolf, Theo Geisel**
Institut für Theoretische Physik and SFB 185 Nichtlineare Dynamik,
Universität Frankfurt, D-60054 Frankfurt/M., and
MPI für Strömungsforschung, D-37018 Göttingen, Germany
email: {klaus,udo,fred,geisel}@chaos.uni-frankfurt.de

## Abstract

Recently Sillito and coworkers (Nature **378**, pp. 492, 1995) demonstrated that stimulation beyond the classical receptive field (cRF) can not only modulate, but radically change a neuron's response to oriented stimuli. They revealed that patch–suppressed cells when stimulated with contrasting orientations inside and outside their cRF can strongly respond to stimuli oriented orthogonal to their nominal preferred orientation. Here we analyze the emergence of such complex response patterns in a simple model of primary visual cortex. We show that the observed sensitivity for orientation contrast can be explained by a delicate interplay between local isotropic interactions and patchy long–range connectivity between distant iso–orientation domains. In particular we demonstrate that the observed properties might arise without specific connections between sites with cross-oriented cRFs.

## 1 Introduction

Long range horizontal connections form a ubiquitous structural element of intracortical circuitry. In the primary visual cortex long range horizontal connections extend over distances spanning several hypercolumns and preferentially connect cells of similar orientation preference [1, 2, 3, 4]. Recent evidence suggests that

their physiological effect depends on the level of postsynaptic depolarization; acting exitatory on weakly activated and inhibitory on strongly activated cells [5, 6]. This differential influence possibly underlies perceptual phenomena as 'pop out' and 'fill in' [9]. Previous modeling studies demonstrated that such differential interactions may arise from a single set of long range excitatory connections terminating both on excitatory and inhibitory neurons in a given target column [7, 8]. By and large these results suggest that long range horizontal connections between columns of like stimulus preference provide a central mechanism for the context dependent regulation of activation in cortical networks.

Recent experiments by Sillito et al. suggest, however, that lateral connections in primary visual cortex can also induce more radical changes in receptive field organization [10]. Most importantly this study shows that patch–suppressed cells can respond selectively to orientation contrast between center and surround of a stimulus even if they are centrally stimulated orthogonal to their preferred orientation. Sillito et al. argued, that these response properties require specific connections between orthogonally tuned columns for which, however, presently there is only weak evidence.

Here we demonstrate that such nonclassical receptive field properties might instead arise as an emergent property of the known intracortical circuitry. We investigate a simple model for intracortical activity dynamics driven by weakly orientation tuned afferent excitation. The cortical actitvity dynamics is based on a continous firing rate description and incorporates both a local center–surround type interaction and long range connections between distant columns of like orientation preference. The connections of distant orientation columns are assumed to act either excitatory or inhibitory depending on the activation of their target neurons. It turns out that this set of interactions not only leads to the emergence of patch–suppressed cells, but also that a large fraction of these cells exhibits a selectivity for orientation contrast very similar to the one observed by Sillito et al. .

## 2 Model

Our target is the analysis of basic rate modulations emerging from local and long range feedback interactions in a simple model of visual cortex. It is therefore appropriate to consider a simple rate dynamics $\dot{\mathbf{x}} = -c \cdot \mathbf{x} + F(\mathbf{x})$, where $\mathbf{x} = \{x_i, i = 1...N\}$ are the activation levels of $N$ neurons. $F(\mathbf{x}) = g(I_{mex}(\mathbf{x}) + I_{lat}(\mathbf{x}) + I_{ext})$, where $g(I) = c_0 \cdot (I - I_{thres})$ if $I > I_{thres}$, and $g(I) = 0$ otherwise, denotes the firing rate or gain function in dependence of the input $I$.

The neurons are arranged in a quadratic array representing a small part of the visual cortex. Within this layer, neuron $i$ has a position $\mathbf{r}_i$ and a preferred orientation $\Phi_i \in [0, 180]$. The input to neuron $i$ has three contributions $I_i = I_i^{mex} + I_i^{lat} + I_i^{ext}$. $I_i^{mex} = \epsilon_{mex} \cdot \sum_{j=1}^{N} w_{ij}^{mex} x_j$ is due to a mexican-hat shaped coupling structure with weights $w_{ij}^{mex}$, $I_{lat} = \epsilon_{lat} \cdot w_L(x_i) \cdot \sum_{j=1}^{N} w_{ij}^{lat} x_j$ denotes input from long-range orientation-specific interactions with weights $w_{ij}^{lat}$, and the third term models the orientation dependent external input $I_{ext} = \epsilon_{ext} \cdot I_{0,i}^{ext} \cdot (1 + \eta_i)$, where $\eta_i$ denotes the noise added to the external input. $w_L(x)$ regulates the strength and sign of the long-

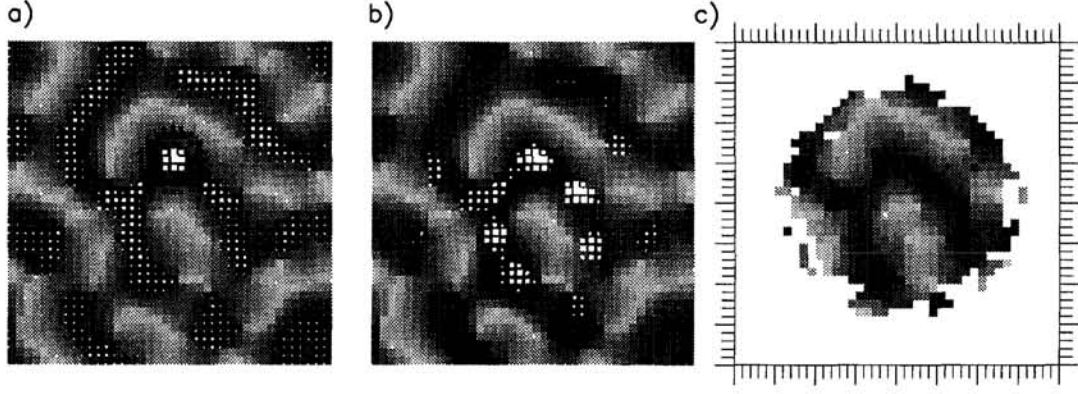

Figure 1: Structure and response properties of the model network. a) Coupling structure from one neuron on a grid of $N = 1600$ elements projected on the orientation preference map which was used for stimulation ($\Phi_i, i = 1...N$). Inhibitory and excitatory couplings are marked with black and white squares, respectively, the sizes of which represent the coupling strength. b) Activation pattern of the network driven by a central stimulus of radius $r_c = 11$ and horizontal orientation. c) Self-consistent orientation map calculated from the activation patterns for all stimulus orientations. Note that this map matches the input orientation preference map shown in a) and b).

range lateral interaction in dependence of the postsynaptic activation, summarizing the behavior of a local circuit in which the inhibitory population has a larger gain.

In particular, $w_l(x)$ can be derived analytically from a simple cortical microcircuit (Fig.2). This circuit consists of an inhibitory and excitatory cell population connected reciprocally. Each population receives lateral input and is driven by the external stimulus $I$. The effective interaction $w_L$ depends on the lateral input $L$ and external input $I$. Assuming a piecewise linear gain function for each population, similar as those for the $x_i$'s, the phase-space $I$-$L$ is partitioned in some regions. Only if both $I$ **and** $L$ are small, $w_L$ is positive; justifying the choice $w_L = x_{sh} - \tanh\left(0.55 \cdot (x - x_a)/x_b\right)$ which we used for our simulations.

The weights $w_{ij}^{mex}$ are given by

$$
\begin{aligned}
w_{ij}^{mex} &= -a_{ex} \cdot \mid \mathbf{r}_i - \mathbf{r}_j \mid^2 + b_{ex} \quad \text{for} \quad \mid \mathbf{r}_i - \mathbf{r}_j \mid \le r_{ex} \\
w_{ij}^{mex} &= a_{in} \cdot \mid \mathbf{r}_i - \mathbf{r}_j \mid^2 - b_{in} \quad \text{for} \quad r_{ex} < \mid \mathbf{r}_i - \mathbf{r}_j \mid \le r_{in}
\end{aligned}
\tag{1}
$$

and $w_{ij}^{mex} = 0$ otherwise. In this representation of the local interactions weights and scales are independently controllable. In particular if we define

$$
a_{ex} = \frac{2}{\pi r_{ex}{}^4}, \; a_{in} = \frac{2 \cdot c_{rel}}{\pi (r_{in} + r_{ex})(r_{in} - r_{ex})^3}, \; b_{ex} = a_{ex} r_{ex}{}^2, \; b_{in} = a_{in}(r_{in} - r_{ex})^2
\tag{2}
$$

$r_{ex}$ and $r_{in}$ denote the range of the excitatory and inhibitory part of the mexican hat, respectively. Here we used $r_{ex} = 2.5$ and $r_{in} = 4.0$. $c_{rel}$ controls the balances of inhibition and excitation. With constant activation level $x_i = x_0 \; \forall i$ the inhibition is $c_{rel}$ times higher than the excitation and $I_{mex} = \epsilon_{mex} \cdot (1 - c_{rel}) \cdot x_0$.

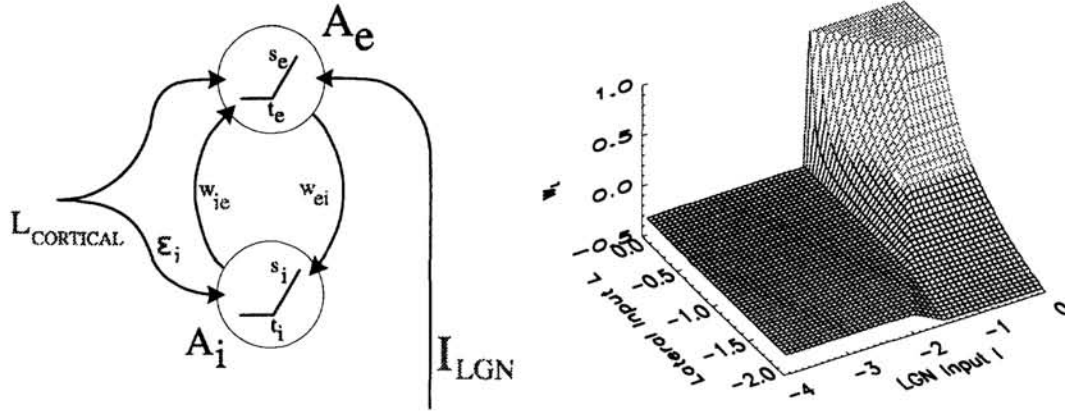

Figure 2: Local cortical circuit (left), consisting of an inhibitory and an excitatory population of neurons interconnected with weights $w_{ie}$, $w_{ei}$, and stimulated with lateral input $L$ and external input $I$. By substituting this circuit with one single excitatory unit, we need a differential instead of a fixed lateral coupling strength ($w_L$, right), which is positive only for small $I$ and $L$.

The weights $w_{ij}^{lat}$ are

$$w_{ij}^{lat} = c_{lat} \exp\left(-\frac{|\Phi_i - \Phi_j|^2}{2\sigma_{lat,\Phi}^2}\right) \exp\left(-\frac{|\mathbf{r}_i - \mathbf{r}_j|^2}{2\sigma_{lat,r}^2}\right) \quad \text{if } |\mathbf{r}_i - \mathbf{r}_j| > r_{in} \quad (3)$$

and 0 otherwise. $\sigma_{lat,\phi}$ and $\sigma_{lat,r}$ provide the orientation selectivity and the range of the long-range lateral interaction, respectively. The additional parameter $c_{lat}$ normalizes $w_{ij}^{lat}$ such that $\sum_{j=1}^{N} w_{ij}^{lat} \approx 1$.

[1]

The spatial width and the orientation selectivity of the input fields are modeled by a convolution with a Gaussian kernel before projected onto the cortical layer

$$I_{0,i}^{ext.} = \frac{1}{2\pi\sigma_{recp,r}^2} \sum_{j=1}^{N} \left[\exp\left(-\frac{|\mathbf{r}_i - \mathbf{r}_j|^2}{2\sigma_{recp,r}^2}\right) \cdot \exp\left(-\frac{|\Phi_i - \Phi_j|^2}{2\sigma_{recp,\Phi}^2}\right)\right].$$

In our simulations, the orientation preference of a cortical neuron $i$ was given by the orientation preference map displayed in Fig1a.

## 3  Results

We analyzed stationary states of our model depending on stimulus conditions. The external input, a center stimulus of radius $r_c = 6$ at orientation $\Phi_c$ and an annulus

a)                          b)                          c)

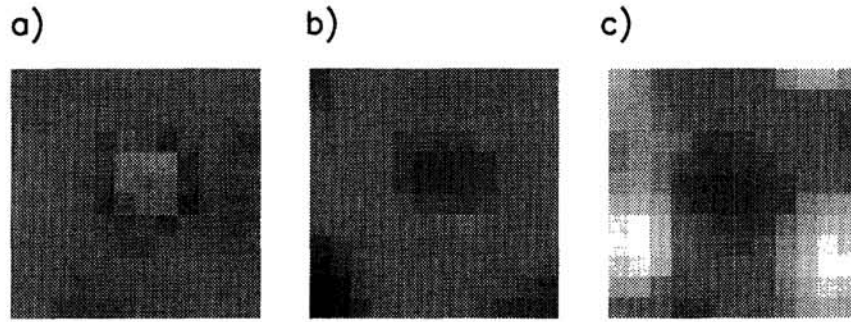

Figure 3: Changes in patterns of activity induced by the additional presentation of a surround stimulus. Grey levels encode increase (darker grey) or decrease (lighter grey) in activation **x**. a) center and surround parallel, b) and c) center and surround orthogonal. While in b), the center is stimulated with the preferred orientation, in c), the center is stimulated with the non-preferred orientation.

of inner radius $r_c$ and outer radius $r_s = 18$ at orientation $\Phi_s$, was projected onto a grid of $N = 40 \times 40$ elements (Fig.1a). Simulations were performed for 20 orientations equally spaced in the interval $[0, 180°]$. When an oriented stimulus was presented to the center we found blob-like activity patterns centered on the corresponding iso–orientation domains (Fig.1b). Simulations utilizing the full set of orientations recovered the input-specificity and demonstrated the self-consistency of the parameter set chosen (Fig.1c). While in this case there were still some deviations, stimulation of the whole field yielded perfect self-consistency (not shown) in the sense of virtually identical afferent and response based orientation preferences.

For combinations of center and surround inputs we observed patch–suppressed regions. These regions exhibited substantial responses for cross–oriented stimuli which often exceeded the response to an optimal center stimulus alone. Figs.3 and 4 summarize these results. Fig.3 shows responses to center–surround combinations compared to activity patterns resulting from center stimulation only. Obviously certain regions within the model were strongly patch–suppressed (Fig.3, light patches for same orientations of center and surround). Interestingly a large fraction of these locations exhibited enhanced activation when center and surround stimulus were orthogonal. Fig.4 displays tuning curves of patch–suppressed cells for variing the orientation of the surround stimulus. Clearly these cells exhibited an enhancement of most responses and a substantial selectivity for orientation contrast. Parameter variation indicated that qualitatively these results do not depend sensitively of the set of parameters chosen.

## 4   Summary and Discussion

Our model implements only elementary assumptions about intracortical interactions. A local sombrero shaped feedback is well known to induce localized blobs of activity with a stereotyped shape [12]. This effect lies at the basis of many models of visual cortex, as e.g. for the explanation of contrast independence of orientation

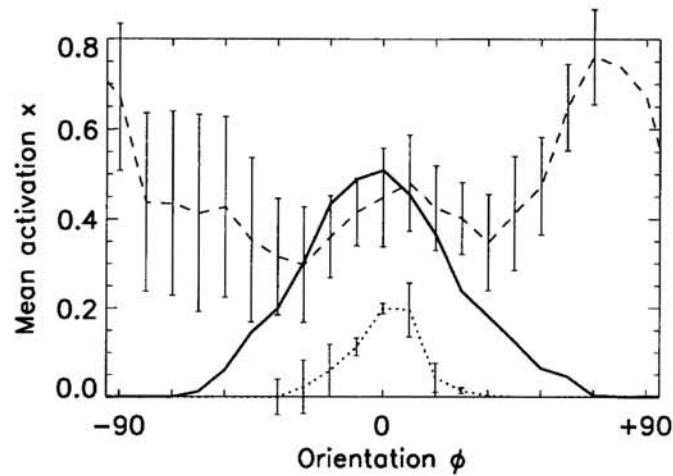

Figure 4: Tuning curves for patch–suppressed cells preferring a horizontal stimulus within their cRF. The bold line shows the orientation tuning curve of the response to an isolated center stimulus. The dashed and dotted lines show the tuning curve when stimulating with a horizontal (dashed) and a vertical (dotted) center stimulus while rotating the surround stimulus. The curves have been averaged over 6 units.

tuning [13, 14]. Long range connections selectively connect columns of similar orientation preference which is consistent with current anatomical knowledge [3, 4]. The differential effect of this set of connections onto the target population was modeled by a continuous sign change of their effective action depending on the level of post-synaptic input or activation. Orientation maps were used to determine the input specificity and we assumed a rather weak selectivity of the afferent connections and a restricted contrast which implies that every stimulus provides some input also to orthogonally tuned cells. This means that long-range excitatory connections, while not effective when only the surround is stimulated, can very well be sufficient for driving cells if the stimulus to the center is orthogonal to their preferred orientation (Contrast sensitivity).

In our model we find a large fraction of cells that exhibit sensitivity for center-surround stimuli. It turns out that most of the patch–suppressed cells respond to orientation contrasts, i.e. they are strongly selective for *orientation discontinuities* between center and surround. We also find contrast enhancement, i.e. larger responses to the preferred orientation in the center when stimulated with an orthogonal surround than if stimulated only centrally (Fig.4). The latter constitutes a genuinely emergent property, since no selective cross–oriented connections are present.

This phenomenon can be understood as a desinhibitory effect. Since no cells having long-range connections to the center unit are activated, the additional sub-threshold input from outside the classical receptive field can evoke a larger response (Contrast enhancement). Contrarily, if center and surround are stimulated with the same ori-

entation, all the cells with similar orientation preference become activated such that the long-range connections can strongly inhibit the center unit (Patch suppression). In other words, while the lack of inhibitory influences from the surround should recover the response with an amplitude similar or higher to the local stimulation, the orthogonal surround effectively leads to a desinhibition for some of the cells.

Our results show a surprising agreement with previous findings on non-classical receptive field properties which culminated in the paper by Sillito et al. [10]. Our simple model clearly demonstrates that the known intracortical interactions might lead to surprising effects on receptive fields. While this contribution concentrated on analyzing the origin of selectivities for orientation discontinuities we expect that the pursued level of abstraction has a large potential for analyzing a wide range of non-classical receptive fields. Despite its simplicity we believe that our model captures the main features of rate interactions. More detailed models based on spiking neurons, however, will exhibit additional dynamical effects like correlations and synchrony which will be at the focus of our future research.

**Acknowledgement**: We acknowledge inspiring discussions with S. Löwel and J. Cowan. This work was supported by the Deutsche Forschungsgemeinschaft.

## Footnotes

[1] The following parameters have been used in our simulations leading to the results shown in Figs.1-4. $\eta_i = 0.1$ (external input noise), $\epsilon_{mex} = 2.2$, $\epsilon_{lat} = 1.5$, $\epsilon_{ext} = 1.3$, $A_{sh} = 0.0$, $A_a = 0.2$, $A_b = 0.05$, $t_e = 0.6$, $s_e = 0.5$, $c_{rel} = 2.0$ (balance between inhibition and excitation), $c_{lat}$ normalizes $w_{ij}^{lat}$ such that $\sum_{j=1}^{N} w_{ij}^{lat} \approx 1$, $\sigma_{lat,\phi} = 20$, $\sigma_{lat,r} = 15$ $\sigma_{recp,r} = 5$, $\sigma_{recp,\Phi} = 40$, $r_{ex} = 2.5$, and $r_{in} = 5.0$.

# References

[1]   D. Ts'o, C.D. Gilbert, and T.N. Wiesel, J. Neurosci **6**, 1160-1170 (1986).

[2]   C.D. Gilbert and T.N. Wiesel, J. Neurosci. **9**, 2432-2442 (1989).

[3]   S. Löwel and W. Singer, Science **255**, 209 (1992).

[4]   R. Malach, Y. Amir, M. Harel, and A. Grinvald, PNAS **90**, 10469-10473 (1993).

[5]   J.A. Hirsch and C.D. Gilbert, J. Neurosci. **6**, 1800-1809 (1991).

[6]   M. Weliky, K. Kandler, D. Fitzpatrick, and L.C. Katz, Neuron **15**, 541-552 (1995).

[7]   M. Stemmler, M. Usher, and E. Niebur, Science **269**, 1877-1880 (1995).

[8]   L.J. Toth, D.C. Sommers, S.C. Rao, E.V. Todorov, D.-S. Kim, S.B. Nelson, A.G. Siapas, and M. Sur, preprint 1995.

[9]   U. Polat, D. Sagi, Vision Res. **7**, 993-999 (1993).

[10]  A.M. Sillito, K.L. Grieve, H.E. Jones, J. Cudeiro, and J. Davis, Nature **378**, 492-496 (1995).

[11]  J.J. Knierim and D.C. van Essen, J. Neurophys. **67**, 961-980 (1992).

[12]  H.R. Wilson and J. Cowan, Biol. Cyb. **13**, 55-80 (1973).

[13]  R. Ben-Yishai, R.L. Bar-Or, and H. Sompolinsky, Proc. Nat. Acad. Sci. **92**, 3844-3848 (1995).

[14]  D. Sommers, S.B. Nelson, and M. Sur, J. Neurosci. **15**, 5448-5465 (1995).